# SCHEMA FOR MOTOR CONTROL

## UTILIZING A NETWORK MODEL OF THE CEREBELLUM

James C. Houk, Ph.D.
Northwestern University Medical School, Chicago, Illinois
60201

## ABSTRACT

This paper outlines a schema for movement control based on two stages of signal processing. The higher stage is a neural network model that treats the cerebellum as an array of adjustable motor pattern generators. This network uses sensory input to preset and to trigger elemental pattern generators and to evaluate their performance. The actual patterned outputs, however, are produced by intrinsic circuitry that includes recurrent loops and is thus capable of self-sustained activity. These patterned outputs are sent as motor commands to local feedback systems called motor servos. The latter control the forces and lengths of individual muscles. Overall control is thus achieved in two stages: (1) an adaptive cerebellar network generates an array of feedforward motor commands and (2) a set of local feedback systems translates these commands into actual movements.

## INTRODUCTION

There is considerable evidence that the cerebellum is involved in the adaptive control of movement[1], although the manner in which this control is achieved is not well understood. As a means of probing these cerebellar mechanisms, my colleagues and I have been conducting microelectrode studies of the neural messages that flow through the intermediate division of the cerebellum and onward to limb muscles via the rubrospinal tract. We regard this cerebellorubrospinal pathway as a useful model system for studying general problems of sensorimotor integration and adaptive brain function. A summary of our findings has been published as a book chapter[2].

On the basis of these and other neurophysiological results, I recently hypothesized that the cerebellum functions as an array of adjustable motor pattern generators[3]. The outputs from these pattern generators are assumed to function as motor commands, i.e., as neural control signals that are sent to lower-level motor systems where they produce movements. According to this hypothesis, the cerebellum uses its extensive sensory input to preset the

pattern generators, to trigger them to initiate the production of patterned outputs and to evaluate the success or failure of the patterns in controlling a motor behavior. However, sensory input appears not to play a major role in shaping the waveforms of the patterned outputs. Instead, these waveforms seem to be produced by intrinsic circuity.

The initial purpose of the present paper is to provide some ideas for a neural network model of the cerebellum that might be capable of accounting for adjustable motor pattern generation. Several previous authors have described network models of the cerebellum that, like the present model, are based on the neuroanatomical organization of this brain structure[4,5,6]. While the present model borrows heavily from these previous models, it has some additional features that may explain the unique manner in which the cerebellum processes sensory input to produce motor commands. A second purpose of this paper is to outline how this network model fits within a broader schema for motor control that I have been developing over the past several years[3,7]. Before presenting these ideas, let me first review some basic physiology and anatomy of the cerebellum[1].

## SIGNALS AND CIRCUITS IN THE CEREBELLUM

There are three main categories of input fibers to the cerebellum, called mossy fibers, climbing fibers and noradrenergic fibers. As illustrated in Fig. 1, the mossy fiber input shows considerable fan-out via granule cells and parallel fibers. The parallel fibers in turn are arranged to provide a high degree of fan-in to individual Purkinje cells (P). These P cells are the sole output elements of the cortical portion of the cerebellum. Via the parallel fiber input, each P cell is exposed to approximately 200,000 potential messages. In marked contrast, the climbing fiber input to P cells is highly focused. Each climbing fiber branches to only 10 P cells, and each cell receives input from only one climbing fiber. Although less is known about input via noradrenergic fibers, it appears to be diffuse and even more divergent than the mossy fiber input.

Mossy fibers originate from several brain sites transmitting a diversity of information about the external world and the internal state of the body. Some mossy fiber inputs are clearly sensory. They come fairly directly from cutaneous, muscle or vestibular receptors. Others are routed via the cerebral cortex where they represent highly processed visual, auditory or somatosensory information. Yet another category of mossy fiber transmits information about central motor commands (Fig. 1 shows one such pathway, from collaterals of the rubrospinal tract relayed

through the lateral reticular nucleus (L)). The discharge rates of mossy fibers are modulated over a wide dynamic range which permits them to transmit detailed parametric information about the state of the body and its external environment.

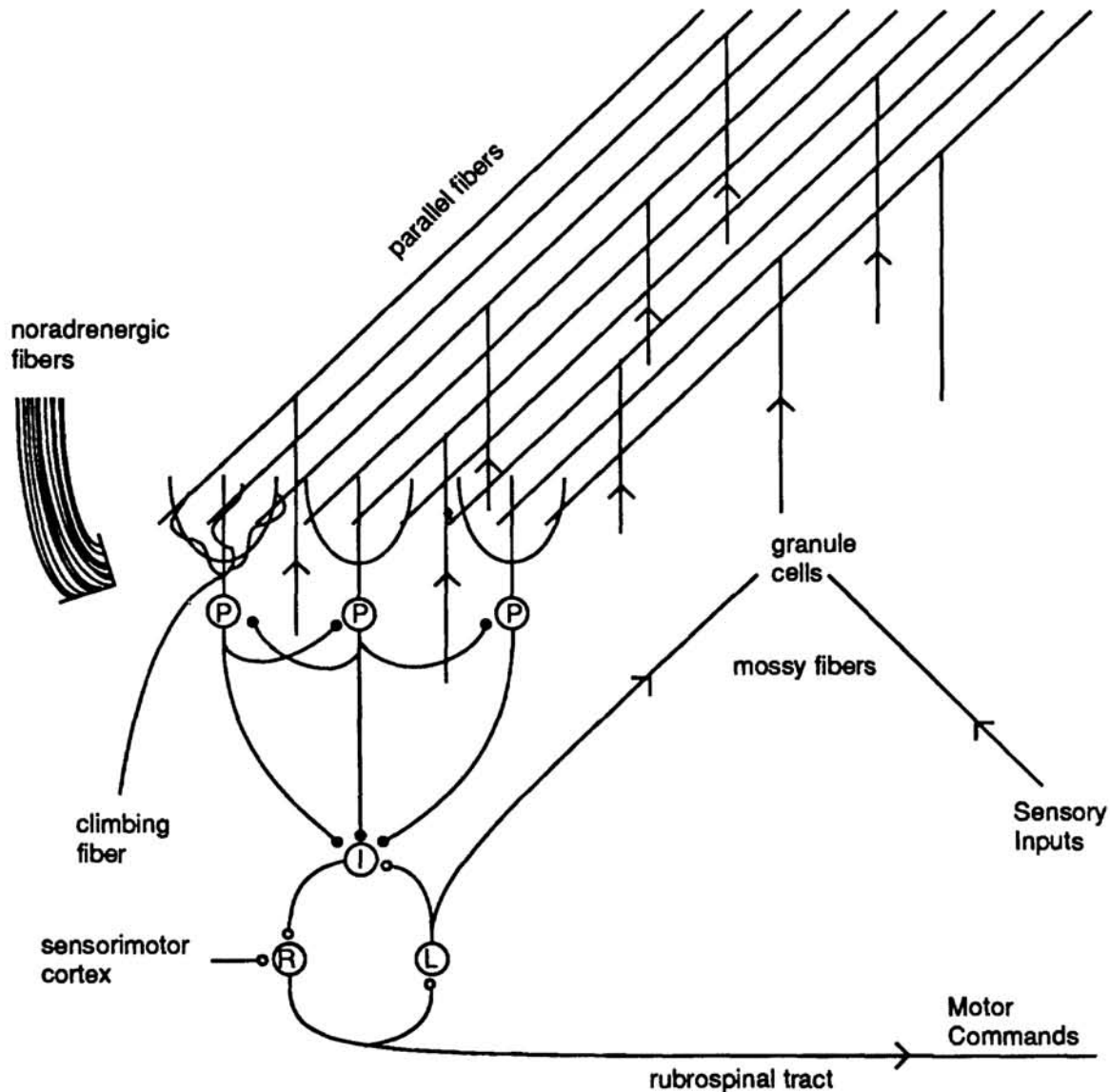

Figure 1: Pathways through the cerebellum. This diagram, which highlights the cerebellorubrospinal system, also constitutes a circuit diagram for the model of an elemental pattern generator.

The sole source of climbing fibers is from cells located in the inferior olivary nucleus. Olivary neurons are selectively sensitive to sensory events. These cells have atypical electrical properties which limit their discharge to rates less than 10 impulses/sec, and usual rates are closer to 1 impulse/sec. As a consequence,

individual climbing fibers transmit very little parametric
information about the intensity and duration of a stimulus;
instead, they appear to be specialized to detect simply the
occurrences of sensory events.  There are also motor inputs
to this pathway, but they appear to be strictly inhibitory.
The motor inputs gate off responsiveness to self-induced
(or expected) stimuli, thus converting olivary neurons into
detectors of unexpected sensory events.

Given the abundance of sensory input to P cells via
mossy and climbing fibers, it is remarkable that these
cells respond so weakly to sensory stimulation.  Instead,
they discharge vigorously during active movements.  P cells
send abundant collaterals to their neighbors, while their
main axons project to the cerebellar nuclei and then onward
to several brain sites that in turn relay motor commands to
the spinal cord.

Fig. 1 shows P cell projections to the intermediate
cerebellar nucleus (I), also called the interpositus
nucleus.  The red nucleus (R) receives its main input from
the interpositus nucleus, and it then transmits motor
commands to the spinal cord via the rubrospinal tract.
Other premotor nuclei that are alternative sources of motor
commands receive input from alternative cerebellar output
circuits.  Fig. 1 thus specifically illustrates the
cerebellorubrospinal system, the portion of the cerebellum
that has been emphasized in my laboratory.

Microelectrode recordings from the red nucleus have
demonstrated signals that appear to represent detailed
velocity commands for distal limb movements.   Bursts of
discharge precede each movement, the frequency of discharge
within the burst corresponds to the velocity of movement,
and the duration of the burst corresponds to the duration
of movement.  These velocity signals are not shaped by
continuous feedback from peripheral receptors; instead,
they appear to be produced centrally.  An important goal of
the modelling effort outlined here is to explain how these
velocity commands might be produced by cerebellar circuits
that function as elemental pattern generators.  I will then
discuss how an array of these pattern generators might
serve well in an overall schema of motor control.

## ELEMENTAL PATTERN GENERATORS

The motivation for proposing pattern generators rather
than more conventional network designs derives from the
experimental observation that motor commands, once initiat-
ed, are not affected, or are only minimally affected, by
alterations in sensory input.  This observation indicates
that the temporal features of these motor commands are
produced by self-sustained activity within the neural
network rather than by the time courses of network inputs.

Two features of the intrinsic circuitry of the cerebellum may be particularly instrumental in explaining self-sustained activity. One is a recurrent pathway from cerebellar nuclei that returns back to cerebellar nuclei. In the case of the cerebellorubrospinal system in Fig. 1, the recurrent pathway is from the interpositus nucleus to red nucleus to lateral reticular nucleus and back to interpositus, what I will call the IRL loop. The other feature of intrinsic cerebellar circuitry that may be of critical importance in pattern generation is mutual inhibition between P cells. Fig. 1 shows how mutual inhibition results from the recurrent collaterals of P-cell axons. Inhibitory interneurons called basket and stellate cells (not shown in Fig. 1) provide additional pathways for mutual inhibition. Both the IRL loop and mutual inhibition between P cells constitute positive feedback circuits and, as such, are capable of self-sustained activity.

Self-sustained activity in the form of high-frequency spontaneous discharge has been observed in the IRL loop under conditions in which the inhibitory P-cell input to I cells is blocked [3]. Trace A in Fig. 2 shows this unrestrained discharge schematically, and the other traces illustrate how a motor command might be sculpted out of this tendency toward high-frequency, repetitive discharge.

Trace B shows a brief burst of input presumed to be sent from the sensorimotor cortex to the R cell in Fig. 1. This burst serves as a trigger that initiates repetitive discharge in an IRL loop, and trace D illustrates the discharge of an I cell in the active loop. The intraburst discharge frequency of this cell is presumed to be determined by the summed magnitude of inhibitory input (shown in trace C) from the set of P cells that project to it (Fig. 1 shows only a few P cells from this set). Since the inhibitory input to I was reduced to an appropriate magnitude for controlling this intraburst frequency some time prior to the arrival of the trigger event, this example illustrates a mechanism for presetting the pattern generator. Note that the same reduction of inhibition that presets the intraburst frequency would bring the loop closer to the threshold for repetitive firing, thus serving to enable the triggering operation. The I-cell burst, after continuing for a duration appropriate for the desired motor behavior, is assumed to be terminated by an abrupt increase in inhibitory input from the set of P cells that project to I (trace C).

The time course of bursting discharge illustrated in Fig. 2D would be expected to propagate throughout the IRL loop and be transmitted via the rubrospinal tract to the spinal cord where it could serve as a motor command. Bursts of R-cell discharge similar to this are observed to precede movements in trained monkey subjects[2].

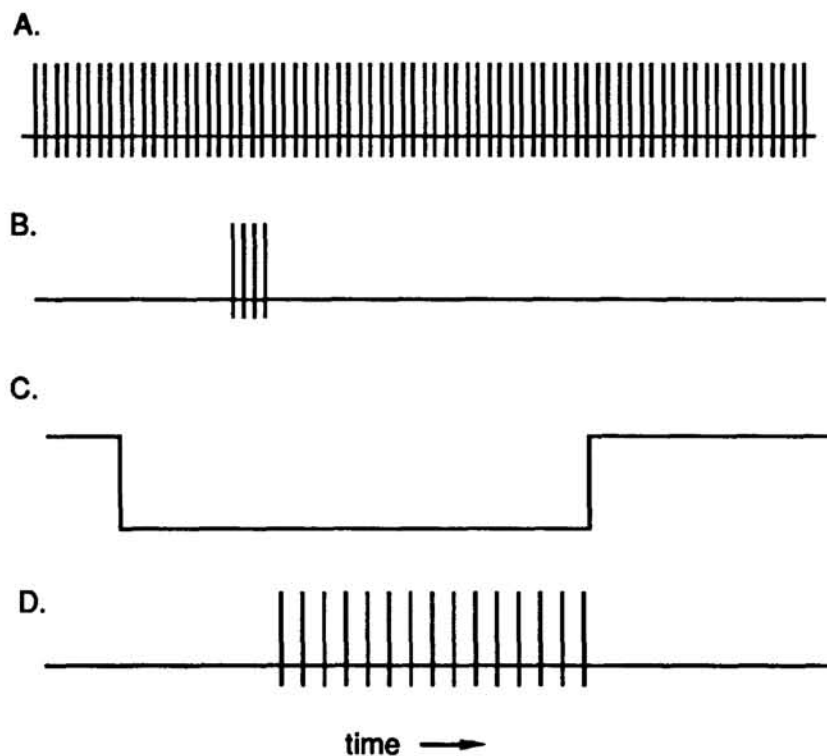

Figure 2: Signals Contributing to Pattern Generation.  A.
    Repetitive discharge of I cell in the absence of P-
    cell inhibition.  B. Trigger burst sent to the IRL
    loop from sensorimotor cortex.  C. Summed inhibition
    produced by the set of P cells projecting to the I
    cell.  D. Resultant motor pattern in I cell.

The sculpting of a motor command out of a repetitive
firing tendency in the IRL loop clearly requires timed
transitions in the discharge rates of specific P cells.
The present model postulates that the latter result from
state transitions in the network of P cells.  Bell and
Grimm[8] described spontaneous transitions in P-cell firing
that occur intermittently, and I have frequently observed
them as well.  These transitions appear to be produced by
intrinsic mechanisms and are difficult to influence with
sensory stimulation.  The mutual recurrent inhibition
between P cells might explain this tendency toward state
transitions.
    Recurrent inhibition between P cells is mediated by
synapses near the cell bodies and primary dendrites of the
P cells whereas parallel fiber input extends far out on the
dendritic tree.  This arrangement may explain why sensory
input via parallel fibers does not have a strong, continu-
ous effect on P cell discharge.  This sensory input may
serve mainly to promote state transitions in the network of
P cells, perhaps by modulating the likelihood that a given
P cell would participate in a state transition.  Once the

transition starts, the activity of the P cell may be domi-
nated by the recurrent inhibition close to the cell body.

The mechanism responsible for the adaptive adjustment
of these elemental pattern generators may be a change in
the synaptic strengths of parallel fiber input to P cells[9].
Such alterations in the efficacy of sensory input would
influence the state transitions discussed in the previous
paragraph, thus mediating adaptive adjustments in the
amplitude and timing of patterned output.  Elsewhere I have
suggested that this learning process is analogous to oper-
ant conditioning and includes both positive and negative
reinforcement[3].  Noradrenergic fibers might mediate posi-
tive reinforcement, whereas climbing fibers might mediate
negative reinforcement.  For example, if the network were
controlling a limb movement, negative reinforcement might
occur when the limb bumps into an object in the work space
(climbing fibers fire in response to unexpected somatic
events such as this), whereas positive reinforcement might
occur whenever the limb successfully acquires the desired
target (the noradrenergic fibers to the cerebellum are
thought to receive input from reward centers in the brain).
Positive reinforcement may be analogous to the associative
reward-punishment algorithm described by Barto[10] which
would fit with the diffuse projections of noradrenergic
fibers.  Negative reinforcement might be capable of a
higher degree of credit assignment in view of the more
focused projections of climbing fibers.

In summary, the previous paragraphs outline some ideas
that may be useful in developing a network model of the
cerebellum.  This particular set of ideas was motivated by
a desire to explain the unique manner in which the cerebel-
lum uses sensory input to control patterned output.  The
model deals explicitly with small circuits within a much
larger network.  The small circuits are considered elemen-
tal pattern generators, whereas the larger network can be
considered an array of these pattern generators.  The
assembly of many elements into an array may give rise to
some emergent properties of the network, due to
interactions between the elements.  However, the highly
compartmentalized anatomical structure of the cerebellum
fosters the notion of relatively independent elemental
pattern generators as hypothesized in the schema for
movement control presented in the next section.

## SCHEMA FOR MOTOR CONTROL

A major aim in developing the elemental pattern
generator model described in the previous section was to
explain the intriguing manner in which the cerebellum uses
sensory input.  Stated succinctly, sensory input is used to
preset and to trigger each elemental pattern generator and

to evaluate the success of previous output patterns in
controlling motor behavior. However, sensory input is not
used to shape the waveform of an ongoing output pattern.
This means that continuous feedback is not available, at
the level of the cerebellum, for any immediate adjustments
of motor commands.

Is this kind of behavior actually advantageous in the
control of movement? I would propose the affirmative,
particularly on the grounds that this strategy seems to
have withstood the test of evolution. Elsewhere I have
reviewed the global strategies that are used to control
several different types of body function[11]. A common
theme in each of these physiological control systems is the
use of negative feedback only as a low-level strategy, and
this coupled with a high-level stage of adaptive
feedforward control. It was argued that this particular
two-stage control strategy is well suited for utilizing the
advantageous features of feedback, feedforward and adaptive
control in combination.

The adjustable pattern generator model of the cerebel-
lum outlined in the previous section is a prime example of
an adaptive, feedforward controller. In the subsequent
paragraphs I will outline how this high-level feedforward
controller communicates with low-level feedback systems
called motor servos to produce limb movements (Fig. 3).

The array of adjustable pattern generators ($PG_n$) in
the first column of Fig. 3 produce an array of elemental
commands that are transmitted via descending fibers to the
spinal cord. The connectivity matrix for descending fibers
represents the consequences of their branching patterns.
Any given fiber is likely to branch to innervate several
motor servos. Similarly, each member of the array of motor
servos ($MS_m$) receives convergent input from a large number
of pattern generators, and the summed total of this input
constitutes its overall motor command.

A motor servo consists of a muscle, its stretch recep-
tors and the spinal reflex pathways back to the same mus-
cle[12]. These reflex pathways constitute negative feedback
loops that interact with the motor command to control the
discharge of the motor neuron pool innervating the
particular muscle. Negative feedback from the muscle
receptors functions to maintain the stiffness of the muscle
relatively constant, thus providing a spring-like interface
between the body and its mechanical environment[13]. The
motor command acts to set the slack length of this
equivalent spring and, in this way, influences motion of
the limb. Feedback also gives rise to an unusual type of
damping proportional to a low fractional power of
velocity[14]. The individual motor servos interact with each
other and with external loads via the trigonometric
relations of the musculoskeletal matrix to produce
resultant joint positions.

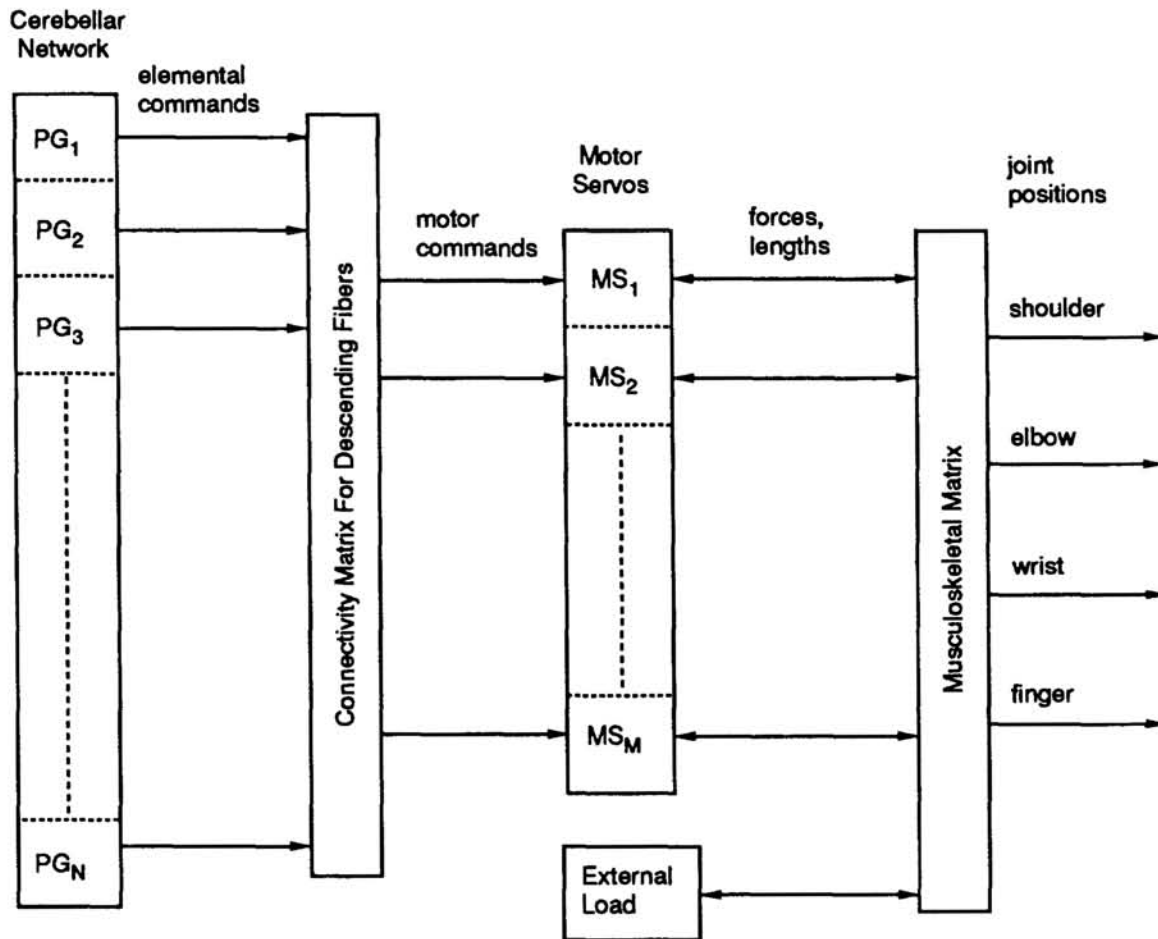

Figure 3: Schema for Motor Control Utilizing Pattern Gen-
erator Model of Cerebellum.  An array of elemental
pattern generators (PG$_n$) operate in an adaptive, feed-
forward manner to produce motor commands.  These out-
puts of the high-level stage are sent to the spinal
cord where they serve as inputs to a low-level array
of negative feedback systems called motor servos
(MS$_m$).  The latter regulate the forces and lengths of
individual muscles to control joint angles.

While the schema for motor control presented here is
based on a considerable body of experimental data, and it
also seems plausible as a strategy for motor control, it
will be important to explore its capabilities for human
limb control with simulation studies.  It may also be
fruitful to apply this schema to problems in robotics.
Since I am mainly an experimentalist, my authorship of this
paper is meant as an entré for collaborative work with
neural network modelers that may be interested in these
problems.

## REFERENCES

1. M. Ito, The Cerebellum and Neural Control (Raven Press, N. Y., 1984).
2. J. C. Houk & A. R. Gibson, In: J. S. King, New Concepts in Cerebellar Neurobiology (Alan R. Liss, Inc., N. Y., 1987), p. 387.
3. J. C. Houk, In: M. Glickstein & C. Yeo, Cerebellum and Neuronal Plasticity (Plenum Press, N. Y., 1988), in press.
4. D. Marr, J. Physiol. (London) 202, 437 (1969).
5. J. S. Albus, Math. Biosci. 10, 25 (1971).
6. C. C. Boylls, A Theory of Cerebellar Function with Applications to Locomotion (COINS Tech. Rep., U. Mass. Amherst), 76-1.
7. J. C. Houk, In: J. E. Desmedt, Cerebral Motor Control in Man: Long Loop Mechanisms (Karger, Basel, 1978), p. 193.
8. C. C. Bell & R. J. Grimm, J. Neurophysiol., 32, 1044 (1969).
9 C.-F. Ekerot & M. Kano, Brain Res., 342, 357 (1985).
10. A. G. Barto, Human Neurobiol., 4, 229 (1985).
11. J. C. Houk, FASEB J., 2, 97-107 (1988).
12. J. C. Houk & W. Z. Rymer, In: V. B. Brooks, Handbook of Physiology, Vol. 1 of Sect. 1 (American Physiological Society, Bethesda, 1981), p.257.
13. J. C. Houk, Annu. Rev. Physiol., 41, 99 (1979).
14. C. C. A. M. Gielen & J. C. Houk, Biol. Cybern., 57, 217 (1987).
